# Exact and Stable Recovery of Sequences of Signals with Sparse Increments via Differential $\ell_1$-Minimization

**Demba Ba[1,2], Behtash Babadi[1,2], Patrick Purdon[2] and Emery Brown[1,2]**
[1]MIT Department of BCS, Cambridge, MA 02139
[2]MGH Department of Anesthesia, Critical Care and Pain Medicine
55 Fruit st, GRJ 4, Boston, MA 02114
demba@mit.edu, {behtash,patrickp}@nmr.mgh.harvard.edu
enb@neurostat.mit.edu

## Abstract

We consider the problem of recovering a sequence of vectors, $(x_k)_{k=0}^{K}$, for which the increments $x_k - x_{k-1}$ are $S_k$-sparse (with $S_k$ typically smaller than $S_1$), based on linear measurements $(y_k = A_k x_k + e_k)_{k=1}^{K}$, where $A_k$ and $e_k$ denote the measurement matrix and noise, respectively. Assuming each $A_k$ obeys the restricted isometry property (RIP) of a certain order—depending only on $S_k$—we show that in the absence of noise a convex program, which minimizes the weighted sum of the $\ell_1$-norm of successive differences subject to the linear measurement constraints, recovers the sequence $(x_k)_{k=1}^{K}$ *exactly*. This is an interesting result because this convex program is equivalent to a standard compressive sensing problem with a highly-structured aggregate measurement matrix which does not satisfy the RIP requirements in the standard sense, and yet we can achieve exact recovery. In the presence of bounded noise, we propose a quadratically-constrained convex program for recovery and derive bounds on the reconstruction error of the sequence. We supplement our theoretical analysis with simulations and an application to real video data. These further support the validity of the proposed approach for acquisition and recovery of signals with time-varying sparsity.

## 1   Introduction

In the field of theoretical signal processing, compressive sensing (CS) has arguably been one of the major developments of the past decade. This claim is supported in part by the deluge of research efforts (see for example Rice University's CS repository [1]) which has followed the inception of this field [2, 3, 4]. CS considers the problem of acquiring and recovering signals that are sparse (or compressible) in a given basis using non-adaptive linear measurements, at a rate smaller than what the Shannon-Nyquist theorem would require. The work [2, 4] derived conditions under which a sparse signal can be recovered *exactly* from a small set of non-adaptive linear measurements. In [3], the authors propose a recovery algorithm for the case of measurements contaminated by bounded noise. They show that this algorithm is stable, that is, within a constant of the noise tolerance. Recovery of these sparse or compressible signals is performed using convex optimization techniques.

The classic CS setting does not take into account the structure, e.g. temporal or spatial, of the underlying high-dimensional sparse signals of interest. In recent years, the attention has shifted to formulations which incorporate the signal structure into the CS framework. A number of problems and applications of interest deal with time-varying signals which may not only be sparse at any given instant, but may also exhibit sparse changes from one instant to the next. For example, a video

of a natural scene consists of a sequence of natural images (compressible signals) which exhibits sparse changes from one frame to the next. It is thus reasonable to hope that one would be able to get away with far fewer measurements than prescribed by conventional CS theory to acquire and recover such time-varying signals as videos. The problem of recovering signals with time-varying sparsity has been referred to in the literature as dynamic CS. A number of *empirically-motivated* algorithms to solve the dynamic CS problem have been proposed, e.g. [5, 6]. To our knowledge, no recovery guarantees have been proved for these algorithms, which typically assume that the support of the signal and/or the amplitudes of the coefficients change smoothly with time. In [5], for instance, the authors propose message-passing algorithms for tracking and smoothing of signals with time-varying sparsity. Simulation results show the superiority of the algorithms compared to one based on applying conventional CS principles at each time instant. Dynamic CS algorithms have potential applications to video processing [7], estimation of sources of brain activity from MEG time-series [8], medical imaging [7], and estimation of time-varying networks [9].

To the best of our knowledge, the dynamic CS problem has not received rigorous, theoretical scrutiny. In this paper, we develop rigorous results for dynamic CS both in the absence and in the presence of noise. More specifically, in the absence of noise, we show that one can *exactly* recover a sequence $(x_k)_{k=0}^K$ of vectors, for which the increments $x_k - x_{k-1}$ are $S_k$-sparse, based on linear measurements $y_k = A_k x_k$ and under certain regularity conditions on $(A_k)_{k=1}^K$, by solving a convex program which minimizes the weighted sum of the $\ell_1$-norms of successive differences. In the presence of noise, we derive error bounds for a quadratically-constrained convex program for recovery of the sequence $(x_k)_{k=0}^K$.

In the following section, we formulate the problem of interest and introduce our notation. In Section 3, we present our main theoretical results, which we supplement with simulated experiments and an application to real video data in Section 4. In this latter section, we introduce probability-of-recovery surfaces for the dynamic CS problem, which generalize the traditional recovery curves of CS. We give concluding remarks in Section 5.

## 2    Problem Formulation and Notation

We denote the support of a vector $x \in \mathbb{R}^p$ by $\text{supp}(x) = \{j : x_j \neq 0\}$. We say that a vector $x \in \mathbb{R}^p$ is $S$-sparse if $||x||_0 \leq S$, where $||x||_0 := |\text{supp}(x)|$. We consider the problem of recovering a sequence $(x_k)_{k=0}^K$ of $\mathbb{R}^p$ vectors such that $x_k - x_{k-1}$ is $S_k$-sparse based on linear measurements of the form $y_k = A_k x_k + e_k$. Here, $A_k \in \mathbb{R}^{n_k \times p}$, $e_k \in \mathbb{R}^{n_k}$ and $y_k \in \mathbb{R}^{n_k}$ denote the measurement matrix, measurement noise, and the observation vector, respectively. Typically, $S_k < n_k \ll p$, which accounts for the compressive nature of the measurements. For convenience, we let $x_0$ be the $\mathbb{R}^p$ vector of all zeros.

For the rest of our treatment, it will be useful to introduce some notation. We will be dealing with sequences (of sets, matrices, vectors), as such we let the index $k$ denote the $k^{\text{th}}$ element of any such sequence. Let $J$ be the set of indices $\{1, 2, \cdots, p\}$. For each $k$, we denote by $\{a_{kj} : j \in J\}$, the columns of the matrix $A_k$ and by $\mathscr{H}_k$ the Hilbert space spanned by these vectors.

For two matrices $A_1 \in \mathbb{R}^{n_1 \times p}$ and $A_2 \in \mathbb{R}^{n_2 \times p}$, $n_2 \leq n_1$, we say that $A_2 \subset A_1$ if the rows of $A_2$ are distinct and each row of $A_2$ coincides with a row of $A_1$.

We say that the matrix $A \in \mathbb{R}^{n \times p}$ satisfies the restricted isometry property (RIP) or order $S$ if, for all $S$-sparse $x \in \mathbb{R}^p$, we have

$$(1 - \delta_S) ||x||_2^2 \leq ||Ax||_2^2 \leq (1 + \delta_S) ||x||_2^2, \tag{1}$$

where $\delta_S \in (0, 1)$ is the smallest constant for which Equation 1 is satisfied [2].

Consider the following convex optimization programs

$$\min_{x_1, x_2, \cdots, x_K} \sum_{k=1}^K \frac{||x_k - x_{k-1}||_1}{\sqrt{S_k}} \qquad s.t. \qquad y_k = A_k x_k, \quad k = 1, 2, \cdots, K. \tag{P1}$$

$$\min_{x_1, x_2, \cdots, x_K} \sum_{k=1}^K \frac{||x_k - x_{k-1}||_1}{\sqrt{S_k}} \qquad s.t. \quad ||y_k - A_k x_k||_2 \leq \epsilon_k, \quad k = 1, 2, \cdots, K. \tag{P2}$$

What theoretical guarantees can we provide on the performance of the above programs for recovery of sequences of signals with sparse increments, respectively in the absence (P1) and in the presence (P2) of noise?

## 3 Theoretical Results

We first present a lemma giving sufficient conditions for the uniqueness of sequences of vectors with sparse increments given linear measurements in the absence of noise. Then, we prove a theorem which shows that, by strengthening the conditions of this lemma, program (P1) can *exactly* recover every sequence of vectors with sparse increments. Finally, we derive error bounds for program (P2) in the context of recovery of sequences of vectors with sparse increments in the presence of noise.

**Lemma 1** (Uniqueness of Sequences of Vectors with Sparse Increments).

*Suppose $(S_k)_{k=0}^K$ is such that $S_0 = 0$, and for each $k$, $S_k \geq 1$. Let $A_k$ satisfy the RIP of order $2S_k$. Let $x_k \in \mathbb{R}^p$ supported on $T_k \subseteq J$ be such that $||x_k - x_{k-1}||_0 \leq S_k$, for $k = 1, 2, \cdots, K$. Suppose $T_0 = \emptyset$ without loss of generality (w.l.o.g.). Then, given $A_k$ and $y_k = A_k x_k$, the sequence of sets $(T_k)_{k=1}^K$, and consequently the sequence of coefficients $(x_k)_{k=1}^K$, can be reconstructed uniquely.*

*Proof.* For brevity, and w.l.o.g., we prove the lemma for $K = 2$. We prove that there is a unique choice of $x_1$ and $x_2$ such that $||x_1 - x_0||_0 \leq S_1$, $||x_2 - x_1||_0 \leq S_2$ and obeying $y_1 = A_1 x_1$, $y_2 = A_2 x_2$. We proceed by contradiction , and assume that there exist $x_1' \neq x_1$ and $x_2' \neq x_2$ supported on $T_1'$ and $T_2'$, respectively, such that $y_1 = A_1 x_1 = A_1 x_1'$, $y_2 = A_2 x_2 = A_2 x_2'$, $||x_1' - x_0||_0 \leq S_1$, and $||x_2' - x_1'||_0 \leq S_2$. Then $||A_1(x_1 - x_1')||_2 = 0$. Using the lower bound in the RIP of $A_1$ and the fact that $\delta_{2S_1} < 1$, this leads to $||x_1 - x_1'||_2^2 = 0$, i.e. $x_1 = x_1'$, thus contradicting our assumption that $x_1 \neq x_1'$. Now consider the case of $x_2$ and $x_2'$. We have

$$0 = A_2(x_2 - x_2') = A_2(x_2 - x_1 + x_1 - x_2') = A_2(x_2 - x_1 + x_1' - x_2'). \qquad (2)$$

Using the lower bound in the RIP of $A_2$ and the fact that $\delta_{2S_2} < 1$, this leads to $||x_2 - x_1 + x_1' - x_2'||_2^2 = 0$, i.e. $x_2 - x_1 = x_2' - x_1'$, which implies $x_2' = x_2$, thus contradicting our assumption that $x_2 \neq x_2'$. $\qquad \square$

As in Candès and Tao's work [2], this lemma only suggests what may be possible in terms of recovery of $(x_k)_{k=1}^K$ through a combinatorial, brute-force approach. By imposing stricter conditions on $(\delta_{2S_k})_{k=1}^K$, we can recover $(x_k)_{k=1}^K$ by solving a convex program. This is summarized in the following theorem.

**Theorem 2** (Exact Recovery in the Absence of Noise).

*Let $(\bar{x}_k)_{k=1}^K \in \mathbb{R}^p$ be a sequence of $\mathbb{R}^p$ vectors such that, for each $k$, $||\bar{x}_k - \bar{x}_{k-1}||_0 \leq S_k$ for some $S_k < p/2$. Suppose that the measurements $y_k = A_k \bar{x}_k \in \mathbb{R}^{n_k}$ are given, such that $n_k < p$, $A_1 \supset A_2$, $A_k = A_2$ for $k = 3, \cdots, K$ and $(A_k)_{k=1}^K$ satisfies $\delta_{S_k} + \delta_{2S_k} + \delta_{3S_k} < 1$ for $k = 1, 2 \cdots, K$. Then, the sequence $(\bar{x}_k)_{k=1}^K$ is the unique minimizer to the program (P1).*

*Proof.* As before, we consider the case $K = 2$. The proof easily generalizes to the case of arbitrary $K$. We can re-write the program as follows:

$$\min_{x_1, x_2} \frac{||x_1||_1}{\sqrt{S_1}} + \frac{||x_2 - x_1||_1}{\sqrt{S_2}} \qquad s.t. \qquad A_1 x_1 = A_1 \bar{x}_1, A_2(x_2 - x_1) = A_2(\bar{x}_2 - \bar{x}_1), \quad (3)$$

where we have used the fact that $A_1 \supset A_2$: $A_2 x_2 - A_1 x_1 = A_2 \bar{x}_2 - A_1 \bar{x}_1$, which implies $A_2(x_2 - x_1) = A_2(\bar{x}_2 - \bar{x}_1)$.

Let $x_1^*$ and $x_2^*$ be the solutions to the above program. Let $T_1 = \text{supp}(\bar{x}_1)$ and $\Delta T_2 = \text{supp}(\bar{x}_2 - \bar{x}_1)$. Assume $|T_1| \leq S_1$ and $|\Delta T_2| \leq S_2$.

**Key element of the proof:** The key element of the proof is the existence of vectors $u_1, u_2$ satisfying the exact reconstruction property (ERP) [10, 11]. It has been shown in [10] that given $\delta_{S_k} + \delta_{2S_k} + \delta_{3S_k} < 1$ for $k = 1, 2$:

1. $\langle u_1, a_{1j} \rangle = \mathrm{sgn}(x_{1,j})$, for all $j \in T_1$, and $\langle u_2, a_{2j} \rangle = \mathrm{sgn}(x_{2,j})$, for all $j \in \Delta T_2$.

2. $|\langle u_1, a_{1j} \rangle| < 1$, for all $j \in T_1^c$, and $|\langle u_2, a_{2j} \rangle| < 1$, for all $j \in \Delta T_2^c$.

Since $\bar{x}_1$ and $\bar{x}_2 - \bar{x}_1$ are feasible, we have

$$\frac{||x_1^*||_1}{\sqrt{S_1}} + \frac{||x_2^* - x_1^*||_1}{\sqrt{S_2}} \leq \frac{||\bar{x}_1||_1}{\sqrt{S_1}} + \frac{||\bar{x}_2 - \bar{x}_1||_1}{\sqrt{S_2}}. \tag{4}$$

$$
\begin{aligned}
\frac{||x_1^*||_1}{\sqrt{S_1}} \;+\; & \frac{||x_2^* - x_1^*||_1}{\sqrt{S_2}} = \frac{1}{\sqrt{S_1}} \sum_{j \in T_1} |\bar{x}_{1,j} + (x_{1,j}^* - \bar{x}_{1,j})| + \frac{1}{\sqrt{S_1}} \sum_{j \in T_1^c} |x_{1,j}^*| \\
+ \quad & \frac{1}{\sqrt{S_2}} \sum_{j \in \Delta T_2} |\bar{x}_{2,j} - \bar{x}_{1,j} + (x_{2,j}^* - x_{1,j}^* - (\bar{x}_{2,j} - \bar{x}_{1,j}))| + \frac{1}{\sqrt{S_2}} \sum_{j \in \Delta T_2^c} |x_{2,j}^* - x_{1,j}^*| \\
\geq \quad & \frac{1}{\sqrt{S_1}} \sum_{j \in T_1} \underbrace{\mathrm{sgn}(\bar{x}_{1,j})}_{\langle u_1, a_{1j} \rangle} (\bar{x}_{1,j} + (x_{1,j}^* - \bar{x}_{1,j})) + \frac{1}{\sqrt{S_1}} \sum_{j \in T_1^c} x_{1,j}^* \langle u_1, a_{1j} \rangle \\
+ \quad & \frac{1}{\sqrt{S_2}} \sum_{j \in \Delta T_2} \underbrace{\mathrm{sgn}(\bar{x}_{2,j} - \bar{x}_{1,j})}_{\langle u_2, a_{2j} \rangle} (\bar{x}_{2,j} - \bar{x}_{1,j} + (x_{2,j}^* - x_{1,j}^* - (\bar{x}_{2,j} - \bar{x}_{1,j}))) \\
+ \quad & \frac{1}{\sqrt{S_2}} \sum_{j \in \Delta T_2^c} (x_{2,j}^* - x_{1,j}^*) \langle u_2, a_{2j} \rangle \\
= \quad & \frac{1}{\sqrt{S_1}} \sum_{j \in T_1} |\bar{x}_{1,j}| + \frac{1}{\sqrt{S_1}} \langle u_1, \underbrace{\sum_{j \in J} x_{1,j}^* a_{1j}}_{A_1 x_1^*} - \underbrace{\sum_{j \in T_1} \bar{x}_{1,j} a_{1j}}_{A_1 \bar{x}_1} \rangle \\
+ \quad & \frac{1}{\sqrt{S_2}} \sum_{j \in \Delta T_2} |\bar{x}_{2,j} - \bar{x}_{1,j}| + \frac{1}{\sqrt{S_2}} \langle u_2, \underbrace{\sum_{j \in J} (x_{2,j}^* - x_{1,j}^*) a_{2j}}_{A_2(x_2^* - x_1^*)} - \underbrace{\sum_{j \in \Delta T_2} (\bar{x}_{2,j} - \bar{x}_{1,j}) a_{2j}}_{A_2(\bar{x}_2 - \bar{x}_1)} \rangle \\
= \quad & \frac{||\bar{x}_1||_1}{\sqrt{S_1}} + \frac{||\bar{x}_2 - \bar{x}_1||_1}{\sqrt{S_2}}. \tag{5}
\end{aligned}
$$

This implies that all of the inequalities in the derivation above must in fact be equalities. In particular,

$$
\begin{aligned}
\frac{1}{\sqrt{S_1}} \sum_{j \in T_1^c} |x_{1,j}^*| \;+\; & \frac{1}{\sqrt{S_2}} \sum_{j \in \Delta T_2^c} |x_{2,j}^* - x_{1,j}^*| \\
= \quad & \frac{1}{\sqrt{S_1}} \sum_{j \in T_1^c} x_{1,j}^* \langle u_1, a_{1j} \rangle + \frac{1}{\sqrt{S_2}} \sum_{j \in \Delta T_2^c} (x_{2,j}^* - x_{1,j}^*) \langle u_2, a_{2j} \rangle \\
\leq \quad & \frac{1}{\sqrt{S_1}} \sum_{j \in T_1^c} |x_{1,j}^*| \underbrace{|\langle u_1, a_{1j} \rangle|}_{<1} + \frac{1}{\sqrt{S_2}} \sum_{j \in \Delta T_2^c} |x_{2,j}^* - x_{1,j}^*| \underbrace{|\langle u_2, a_{2j} \rangle|}_{<1}.
\end{aligned}
$$

Therefore, $x_{1,j}^* = 0 \; \forall j \in T_1^c$, and $x_{2,j}^* - x_{1,j}^* = 0 \; \forall j \in \Delta T_2^c$. Using the lower bounds in the RIP of $A_1$ and $A_2$ leads to

$$0 = ||A_1(x_1^* - \bar{x}_1)||_2 \quad \geq \quad (1 - \delta_{2S_1}) ||x_1^* - \bar{x}_1||_2 \tag{6}$$

$$0 = ||A_2(x_2^* - x_1^* - (\bar{x}_2 - \bar{x}_1))||_2 \quad \geq \quad (1 - \delta_{2S_2}) ||x_2^* - x_1^* - (\bar{x}_2 - \bar{x}_1)||_2, \tag{7}$$

so that $x_1^* = \bar{x}_1$, and $x_2^* = \bar{x}_2$. Uniqueness follows from simple convexity arguments. $\qquad \square$

A few remarks are in order. First, Theorem 2 effectively asserts that the program (P1) is equivalent to sequentially solving (i.e. for $k = 1, 2, \cdots, K$) the following program, starting with $x_0^*$ the vector of all zeros in $\mathbb{R}^p$:

$$\min_{x_k} ||x_k - x_{k-1}^*||_1 \qquad s.t. \qquad y_k - A_k x_{k-1}^* = A_k(x_k - x_{k-1}^*), \quad k = 1, 2, \cdots, K. \tag{8}$$

Second, it is interesting and surprising that Theorem 2 would hold, if one naively applies standard CS principles to our problem. To see this, if we let $w_k = x_k - x_{k-1}$, then program (P1) becomes

$$\min_{w_1, \cdots, w_K} \sum_{k=1}^{K} \frac{\|w_k\|_1}{\sqrt{S_k}} \qquad s.t. \qquad y = Aw, \tag{9}$$

where $w = (w_1', \cdots, w_K')' \in \mathbb{R}^{K \times p}$, $y = (y_1', \cdots, y_K')' \in \mathbb{R}^{\sum_{k=1}^{K} n_k}$ and $A$ is given by

$$A = \left[ \begin{array}{c|c|c|c} A_1 & 0 & \cdots & 0 \\ \hline A_2 & A_2 & \cdots & 0 \\ \hline \vdots & \vdots & \ddots & \vdots \\ \hline A_K & A_K & \cdots & A_K \end{array} \right].$$

As $K$ grows large, the columns of $A$ become increasingly correlated or coherent, which intuitively means that $A$ would be far from satisfying RIP of any order. Yet, we get exact recovery. This is an important reminder that the RIP is a sufficient, but not necessary condition for recovery.

Third, the assumption that $A_1 \supset A_2$, $A_k = A_2$ for $k = 3, \cdots, K$ makes practical sense as it allows one to avoid the prohibitive storage and computational cost of generating several distinct measurement matrices. Note that if a random $A_1$ satisfies the RIP of some order and $A_1 \supset A_2$, then $A_2$ also satisfies the RIP (of lower order).

Lastly, the key advantage of dynamic CS recovery (P1) is the smaller number of measurements required compared to the classical approach [2] which would solve $K$ separate $\ell_1$-minimization problems. For each $k = 1, \cdots, K$, one would require $n_k \geq CS_k \log(p/S_k)$ measurements for dynamic recovery, compared to $n_k \geq CS_1 \log(p/S_1)$ for classical recovery. Due to the hypothesis of $S_k \leq S_1 \ll p$, i.e., the sparse increments are small, we conclude that there are less number of measurements required for dynamic CS.

We now move to the case where the measurements are perturbed by bounded noise. More specifically, we derive error bounds for a quadratically-constrained convex program for recovery of sequences of vectors with sparse increments in the presence of noise.

**Theorem 3** (Conditionally Stable Recovery in Presence of Noise)**.**

*Let $(\bar{x}_k)_{k=1}^{K} \in \mathbb{R}^p$ be as stated in Theorem 2, and $x_0$ be the vector of all zeros in $\mathbb{R}^p$. Suppose that the measurements $y_k = A_k x_k + e_k \in \mathbb{R}^{n_k}$ are given such that $\|e_k\|_2 \leq \epsilon_k$ and $(A_k)_{k=1}^{K}$ satisfy $\delta_{3S_k} + 3\delta_{4S_k} < 2$, for each $k$. Let $(x_k^*)_{k=1}^{K}$ be the solution to the program (P2). Finally, let $h_k := (x_k^* - x_{k-1}^*) - (\bar{x}_k - \bar{x}_{k-1})$, for $k = 1, 2, \cdots, K$, with the convention that $\bar{x}_0 := x_0^* := 0 \in \mathbb{R}^p$. Then, we have:*

$$\sum_{k=1}^{K} \|h_k\|_2 \leq \sum_{k=1}^{K} 2C_{S_k} \epsilon_k + \sum_{k=2}^{K} C_{S_k} \left\| A_k \sum_{\ell < k} h_\ell \right\|_2 \tag{10}$$

*where, for each $k = 1, 2, \cdots, K$, $C_{S_k}$ is only a function of $\delta_{3S_k}$ and $\delta_{4S_k}$.*

*Proof sketch.* Candès et al.'s proof for stable recovery in the presence of bounded noise relies on the so-called tube and cone constraints [3]. Our proof for Theorem 3 relies on generalization of these two constraints. We omit some of the algebraic details of the proof as they can be filled in by following the proof of [3] for the time-invariant case.

**Generalized tube constraint:** Let $\bar{w}_k = \bar{x}_k - \bar{x}_{k-1}$, $w_k^* = x_k^* - x_{k-1}^*$, for $k = 1, \cdots, K$. The generalized tube constraints are obtained using a simple application of the triangle inequality:

$$\|A_1(\bar{w}_1 - w_1^*)\|_2 \quad \leq \quad 2\epsilon_1 \tag{11}$$
$$\|A_2(\bar{w}_2 - w_2^*)\|_2 \quad \leq \quad 2\epsilon_2 + \|A_2 h_1\|_2 \text{ and more generally,} \tag{12}$$
$$\|A_k(\bar{w}_k - w_k^*)\|_2 \quad \leq \quad 2\epsilon_k + \left\| A_k \sum_{\ell < k} h_\ell \right\|_2, \text{ for } k = 2, \cdots, K. \tag{13}$$

**Generalized cone constraint:** To obtain a generalization of the cone constraint in [3], we need to account for the fact that the increments $(x_k - x_{k-1})_{k=1}^K$ (may) have different support sizes. The resulting generalized cone constraint is as follows:

$$\sum_{k=1}^K \frac{||h_{k\Delta T_k^c}||_1}{\sqrt{S_k}} \leq \sum_{k=1}^K \frac{||h_{k\Delta T_k}||_1}{\sqrt{S_k}}, \tag{14}$$

where $\Delta T_k = \text{supp}(\bar{x}_k - \bar{x}_{k-1})$. The proof proceeds along the lines of that presented in [3], with $C_{S_k} = \frac{1+\sqrt{1/3}}{\sqrt{1-\delta_{4S_k}} - \sqrt{\frac{1+\delta_{3S_k}}{3}}}$. $\qquad\qquad\qquad\qquad\qquad\qquad\qquad\qquad\qquad\qquad\quad$ □

Equation (10) is an implicit bound: the second term in the inequality reflects the fact that, for a given $k$, the error $x_k^* - \bar{x}_k$ depends on previous errors. Our bound proves a form of stability that is *conditional* on the stability of previous estimates. The appeal of dynamic CS comes from the fact that one may pick the constants $C_{S_k}$ in the bound above to be much smaller that those from the corresponding conventional CS bound [3] (Equation (10) without the second term). This ensures that the errors do not propagate in an unbounded manner. One may obtain sharper bounds using techniques as in [12]. In the next section, we use simulations to compare explicitly the average mean-squared error (MSE) of conventional CS and our algorithm.

## 4    Experiments/Simulations

We ran a series of numerical experiments to assess the ability of the convex programs introduced to recover signals with time-varying sparsity. In the absence of noise, the experiments result in probability-of-recovery surfaces for the dynamic CS problem, which generalize the traditional recovery curves of CS. In the presence of noise, we compare dynamic CS to conventional CS in terms of their reconstruction error as a function of signal-to-noise-ratio (SNR). We also show an application to real video data. All optimization problems were solved using CVX, a package for specifying and solving convex programs [13, 14].

### 4.1    Simulated noiseless data

**Experimental set-up:**

1. Select $n_k$, for $k = 1, \cdots, K$, and $p$, so that the $A_k$'s are $n_k \times p$ matrices; sample $A_k$ with independent Gaussian entries, for $k = 1, 2, \cdots, K$.
2. Select $S_1 = \lceil s_1 \cdot p \rceil$, $s_1 \in (0, 1)$, and $S_k = \lceil s_2 \cdot p \rceil$, $s_2 \in (0, 1)$, for $k = 2, \cdots, K$.
3. Select $T_1$ of size $S_1$ uniformly at random and set $\bar{x}_{1,j} = 1$ for all $j \in T_1$, and 0 otherwise; for $k = 2, \cdots, K$, select $\Delta T_k = \text{supp}(\bar{x}_k - \bar{x}_{k-1})$ of size $S_k$ uniformly at random and set $\bar{x}_{k,j} - \bar{x}_{k-1,j} = 1$ for all $j \in \Delta T_k$, and 0 otherwise.
4. Make $y_k = A_k \bar{x}_k$, for $k = 1, 2, \cdots, K$; solve the program (P1) to obtain $(x_k^*)_{k=1}^K$.
5. Compare $(\bar{x}_k)_{k=1}^K$ to $(x_k^*)_{k=1}^K$.
6. Repeat 100 times for each $(s_1, s_2)$.

We compare dynamic CS to conventional CS applied independently at each $k$. Figure 1 shows results for $n_k = 100$, $p = 200$, and $K = 2$. We can infer the expected behavior for larger values of $K$ from the case $K = 2$ and from the theory developed above (see remarks below).

The probability of recovery for conventional CS is 1 on the set $\{(s_1, s_2) : s_1 + (K-1)s_2 \leq s^*\}$, and 0 on its complement, where $s^*$ is the sparsity level at which a phase transition occurs in the conventional CS problem [2]. The figure shows that, when the measurement matrices $A_k$, for $k = 2, \cdots, K$ are derived from $A_1$ as assumed in Theorem 1, dynamic CS (DCS 1) outperforms conventional CS (CCS). However, when we used different measurement matrices (DCS 2), we see that there is an asymmetry between $s_1$ and $s_2$, which is not predicted by our Theorem 1. Intuitively, this is because for small $s_2$, the program (P1) operates in a regime where we have not only one but multiple measurements to recover a given sparse vector [15]. Program (P1) is equivalent to sequential CS. Therefore, we expect the behavior of conventional CS to persist for larger $K$.

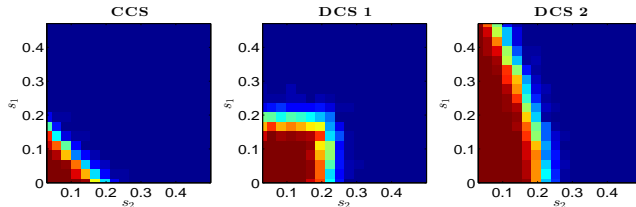

Figure 1: Probability of recovery maps as a function of $s_1$ and $s_2$.

## 4.2 Simulated noisy data

The experimental set-up differs slightly from the one of the noiseless case. In Step 2, we fix constant values for $S_1$ and $S_k$, $k = 2, \cdots, K$. Moreover, in Step 4, we form $y_k = A_k x_k + e_k$, where the $e_k$'s are drawn uniformly in $(-\alpha, \alpha)$. In Step 6, we repeat the experiment 100 times for each $\alpha$. In our experiments, we used $n_1 = 100$, $S_1 = 5$, $n_2 = 20$, $S_k = 1$, for $k = 2, \cdots, K$, and $p = 200$. We report results for $K = 2$ and $K = 10$, and choose values of $\alpha$ resulting in SNRs in the range $[5, 30]$ $dB$, in increments of $5\ dB$.

Figure 2 displays the average MSE given by $10 \cdot \log_{10}(\frac{1}{K} \sum_{k=1}^{K} ||\bar{x}_k - x_k^*||_2^2)$ of conventional CS and dynamic CS as a function of SNR. The Figure shows that the proposed algorithm outperforms conventional CS, and is robust to noise.

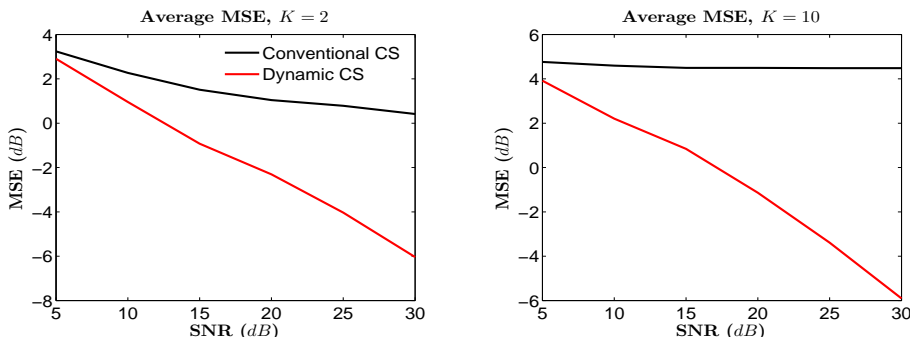

Figure 2: Average MSE as a function of SNR.

## 4.3 Real video data

We consider the problem of recovering the first 10 frames of a real video using our dynamic CS algorithm, and conventional CS applied to each frame separately. In both cases, we assume the absence of noise. We use a video portraying a close-up of a woman engaged in a telephonic conversation [16]. The video has a frame rate of $12\ Hz$ and a total of $150$ frames, each of size $176 \times 144$. Due to computational constraints, we downsampled each frame by a factor of 3 in each dimension. We obtained measurements in the wavelet domain by performing a two-level decomposition of each frame using Daubechies-1 wavelet.

In Table 1, we report the negative of the normalized MSE given by $-10 \cdot \log_{10}(\frac{1}{10} \sum_{k=1}^{10} \frac{||\bar{x}_k - x_k^*||_2^2}{||\bar{x}_k||_2^2})$ in $dB$ for various $(n_1, n_2)$ measurement pairs ($n_k = n_2$, for $k = 3, \cdots, 10$). Larger numbers indicate better reconstruction accuracy. The table shows that, for all $(n_1, n_2)$ considered, dynamic CS outperforms conventional CS. The average performance gap across $(n_1, n_2)$ pairs is approximately $7\ dB$. Interestingly, for sufficient number of measurements, dynamic CS improves as the video progresses. We observed this phenomenon in the small-$s_2$ regime of the simulations. Figure 3 shows the reconstructed frames highlighted in Table 1. The frames reconstructed using dynamic CS are more appealing visually than their conventional CS counterparts.

Table 1: Normalized negated MSE in $dB$ for frames 1, 5, 10, and average over all 10 frames. Each frame consist of $\approx 3000$ pixels. Each row of the table corresponds to a different $(n_1, n_2)$ pair (refer to text). Larger numbers indicate better reconstruction accuracy.

| | Frame 1 | | Frame 5 | | Frame 10 | | Avg. (10 frames) | |
|---|---|---|---|---|---|---|---|---|
| | CCS | DCS | CCS | DCS | CCS | DCS | CCS | DCS |
| (2400,2400) | 27.8 | 27.8 | 28.5 | 38 | 28 | 41.1 | 28.2 | 35 |
| (2000,2000) | **22.4** | **22.4** | **22.3** | **31.3** | **22.9** | **35.6** | 22.8 | 28.9 |
| (2400,1200) | 27.8 | 27.8 | 15.2 | 24.2 | 14.8 | 25.4 | 15.9 | 25.5 |
| (1600,1600) | 19.1 | 19.1 | 18.9 | 25 | 19.8 | 29.7 | 19.1 | 24.1 |
| (1600,800) | 19.2 | 19.2 | 8.4 | 17.6 | 9.3 | 16.7 | 8.4 | 17.8 |

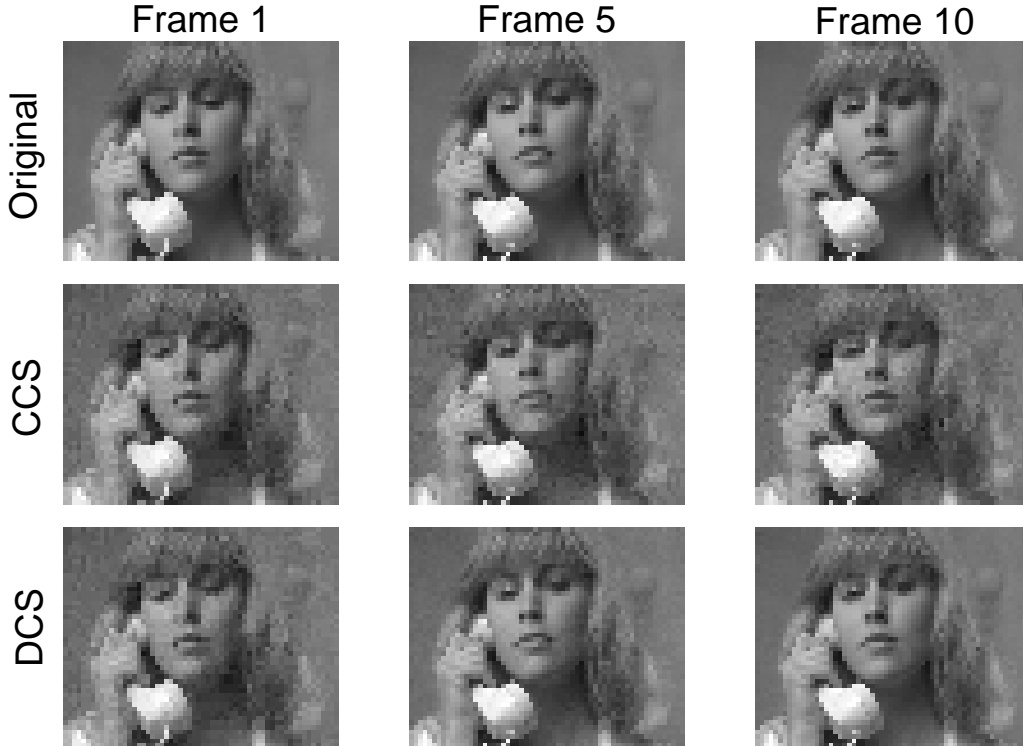

Figure 3: Comparison of frames reconstructed using dynamic CS and conventional CS, $(n_1, n_2) = (2000, 2000)$.

## 5 Discussion

In this paper, we proved rigorous guarantees for convex programs for recovery of sequences of vectors with sparse increments, both in the absence and in the presence of noise. Our formulation of the dynamic CS problem is more general than the empirically-motivated solutions proposed in the literature, e.g. [5, 6]. Indeed, we only require that $x_1$ is sparse, as well as the *increments*. Therefore, there may exist values of $k$ such that $x_k$ is *not* a sparse vector. We supplemented our theoretical analysis with simulation experiments and an application to real video data. In the noiseless case, we introduced probability-of-recovery surfaces which generalize traditional CS recovery curves. The recovery surface showed that dynamic CS significantly outperforms conventional CS, especially for large sequences (large $K$). In the noisy case, simulations showed that dynamic CS also outperforms conventional CS for SNR values ranging from 5 to 30 $dB$. Our results on real video data demonstrated that dynamic CS outperforms conventional CS in terms of visual appeal of the reconstructed frames, and by an average MSE gap of $7dB$.

# References

[1] Compressive sensing resources, rice university. Rice University, `http://dsp.rice/edu/cs/`.

[2] E.J. Candès and T. Tao. Decoding by linear programming. *Information Theory, IEEE Transactions on*, 51(12):4203–4215, 2005.

[3] E.J. Candès, J.K. Romberg, and T. Tao. Stable signal recovery from incomplete and inaccurate measurements. *Communications on pure and applied mathematics*, 59(8):1207–1223, 2006.

[4] D.L. Donoho. Compressed sensing. *Information Theory, IEEE Transactions on*, 52(4):1289–1306, 2006.

[5] J. Ziniel, L.C. Potter, and P. Schniter. Tracking and smoothing of time-varying sparse signals via approximate belief propagation. In *Signals, Systems and Computers (ASILOMAR), 2010 Conference Record of the Forty Fourth Asilomar Conference on*, pages 808–812. IEEE, 2010.

[6] M. Salman Asif and J. Romberg. Dynamic updating for $\ell_1$-minimization. *Selected Topics in Signal Processing, IEEE Journal of*, 4(2):421–434, 2010.

[7] H. Jung and J.C. Ye. Motion estimated and compensated compressed sensing dynamic magnetic resonance imaging: What we can learn from video compression techniques. *International Journal of Imaging Systems and Technology*, 20(2):81–98, 2010.

[8] J.W. Phillips, R.M. Leahy, and J.C. Mosher. Meg-based imaging of focal neuronal current sources. *Medical Imaging, IEEE Transactions on*, 16(3):338–348, 1997.

[9] M. Kolar, L. Song, A. Ahmed, and E.P. Xing. Estimating time-varying networks. *The Annals of Applied Statistics*, 4(1):94–123, 2010.

[10] E. Candès, J. Romberg, and T. Tao. Robust uncertainty principles: Exact signal reconstruction from highly incomplete frequency information. *IEEE Trans. Inform. Theory*, June 2004. Submitted.

[11] E. Candès and T. Tao. Near optimal signal recovery from random projections: Universal encoding strategies? *IEEE Trans. Inform. Theory*, October 2004. Submitted.

[12] E.J. Candès. The restricted isometry property and its implications for compressed sensing. *Comptes Rendus Mathematique*, 346(9):589–592, 2008.

[13] M. Grant and S. Boyd. CVX: Matlab software for disciplined convex programming, version 1.22. `http://cvxr.com/cvx`, May 2012.

[14] M. Grant and S. Boyd. Graph implementations for nonsmooth convex programs. In V. Blondel, S. Boyd, and H. Kimura, editors, *Recent Advances in Learning and Control*, Lecture Notes in Control and Information Sciences, pages 95–110. Springer-Verlag Limited, 2008.

[15] S.F. Cotter, B.D. Rao, K. Engan, and K. Kreutz-Delgado. Sparse solutions to linear inverse problems with multiple measurement vectors. *Signal Processing, IEEE Transactions on*, 53(7):2477–2488, 2005.

[16] Softage video codec demo download page. Softage, `http:www.softage.ru/products/video-codec/uncompressed/suzie.avi`.

